# Analog VLSI Cellular Implementation of the Boundary Contour System

**Gert Cauwenberghs** and **James Waskiewicz**
Department of Electrical and Computer Engineering
Johns Hopkins University
3400 North Charles Street
Baltimore, MD 21218-2686
E-mail: {gert,davros}@bach.ece.jhu.edu

## Abstract

We present an analog VLSI cellular architecture implementing a simplified version of the Boundary Contour System (BCS) for real-time image processing. Inspired by neuromorphic models across several layers of visual cortex, the design integrates in each pixel the functions of simple cells, complex cells, hyper-complex cells, and bipole cells, in three orientations interconnected on a hexagonal grid. Analog current-mode CMOS circuits are used throughout to perform edge detection, local inhibition, directionally selective long-range diffusive kernels, and renormalizing global gain control. Experimental results from a fabricated $12 \times 10$ pixel prototype in $1.2 \ \mu$m CMOS technology demonstrate the robustness of the architecture in selecting image contours in a cluttered and noisy background.

## 1 Introduction

The Boundary Contour System (BCS) and Feature Contour System (FCS) combine models for processes of image segmentation, feature filling, and surface reconstruction in biological vision systems [1],[2]. They provide a powerful technique to recognize patterns and restore image quality under excessive fixed pattern noise, such as in SAR images [3]. A related model with similar functional and structural properties is presented in [4].

The motivation for implementing a relatively complex model such as BCS and FCS on the focal-plane is dual. First, as argued in [5], complex neuromorphic active pixel designs become viable engineering solutions as the feature size of the VLSI technology shrinks significantly below the optical diffraction limit, and more transistors can be stuffed in each pixel. The pixel design that we present contains 88 transistors, likely the most complex

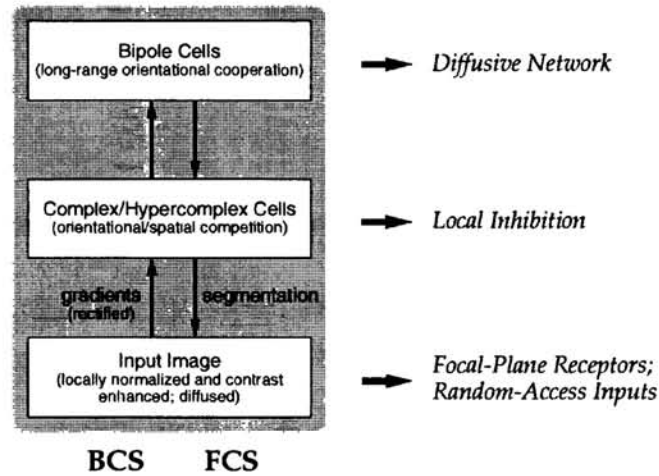

Figure 1: *Diagram of BCS/FCS model for image segmentation, feature filling, and surface reconstruction. Three layers represent simple, complex and bipole cells.*

active pixel imager ever put on silicon. Second, our motivation is to extend the functionality of previous work on analog VLSI neuromorphic image processors for image boundary segmentation, *e.g.* [6, 7, 5, 8, 9] which are based on simplified physical models that do not include directional selectivity and/or long-range signal aggregation for boundary formation in the presence of significant noise and clutter. The analog VLSI implementation of BCS reported here is a first step towards this goal, with the additional objectives of real-time, low-power operation as required for demanding target recognition applications. As an alternative to focal-plane optical input, the image can be loaded electronically through random-access pixel addressing.

The BCS model encompasses visual processing at different levels, including several layers of cells interacting through shunting inhibition, long-range cooperative excitation, and renormalization. The implementation architecture, shown schematically in Figure 1, partitions the BCS model into three levels: simple cells, complex and hypercomplex cells, and bipole cells.

Simple cells compute unidirectional gradients of normalized intensity obtained from the photoreceptors. Complex (hyper-complex) cells perform spatial and directional competition (inhibition) for edge formation. Bipole cells perform long-range cooperation for boundary contour enhancement, and exert positive feedback (excitation) onto the hyper-complex cells. Our present implementation does not include the FCS model, which completes and fills features through diffusive spatial filtering of the image blocked by the edges formed in BCS.

## 2   Modified BCS Algorithm and Implementation

We adopted the BCS algorithm for analog continuous-time implementation on a hexagonal grid, extending in three directions $u$, $v$ and $w$ on the focal plane as indicated schematically in Figure 2. For notational convenience, let *subscript* 0 denote the center pixel and $\pm u$, $\pm v$ and $\pm w$ its six neighbors. Components of each complex cell "vector" $C_i$ at grid location $i$, along three directions of edge selectivity, are indicated with *superscript* indices $u$, $v$ and $w$.

In the implemented circuit model, a pixel unit consists of a photosensor (or random-access analog memory) sourcing a current indicating light intensity, gradient computation and rectification circuits implementing simple cells in three directions, and one complex (hyper-

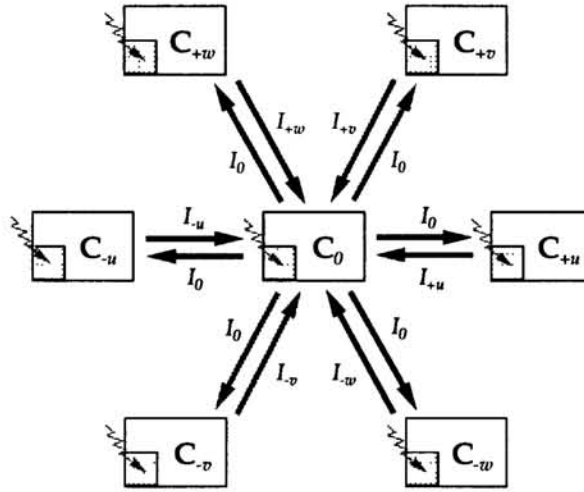

Figure 2: *Hexagonal arrangement of BCS pixels, at the level of simple and complex cells, extending in three directions u, v and w in the focal plane.*

complex) cell and one bipole cell for each of the three directions.

The photosensors generate a current $I_i$ that is proportional to intensity. Through current mirrors, the currents $I_i$ propagate in the three directions $u$, $v$, and $w$ as noted in Figure 2. Rectified finite-difference gradient estimates of $I_i$ are obtained for each of the three hexagonal directions. These gradients excite the complex cells $C_i^j$.

Lateral inhibition among spatially ($i$) and directionally ($j$) adjacent complex cells implement the function of hypercomplex cells for edge enhancement and noise reduction. The complex output ($C_i^j$) is inhibited by local complex cell outputs in the two competing directions of $j$. $C_0$ is additionally inhibited by the complex cells of the four nearest neighbors in competing locations $i$ with parallel orientation.

A directionally selective interconnected diffusive network of bipole cells $B_i^j$, interacting with the complex cells $C_i^j$, provides long range cooperative feedback, and enhances smooth edge contours while reducing spurious edges due to image clutter. $C_i^j$ is excited by bipole interaction received from the bipole cell $B_i^j$ on the line crossing $i$ in the same direction $j$.

The operation of the (hyper-)complex cells in the hexagonal arrangement is summarized in the following equation, for one of the three directions $u$:

$$C_0^u = \left| \frac{1}{2}(I_v + I_w) - I_0 \right| - \alpha(C_0^v + C_0^w) - \alpha'(C_v^u + C_w^u + C_{-v}^u + C_{-w}^u) + \beta B_0^u \quad (1)$$

where:

1. $\left|\frac{1}{2}(I_v + I_w) - I_0\right|$ represents the rectified gradient input as approximated on the hexagonal grid;

2. $\alpha(C_0^v + C_0^w)$ is the inhibition from locally opposing directions;

3. $\alpha'(C_v^u + C_w^u + C_{-v}^u + C_{-w}^u)$ is inhibition from non-aligned neighbors in the same direction; and

4. $\beta B_0^u$ is the excitation through long-range cooperation from the bipole cell.

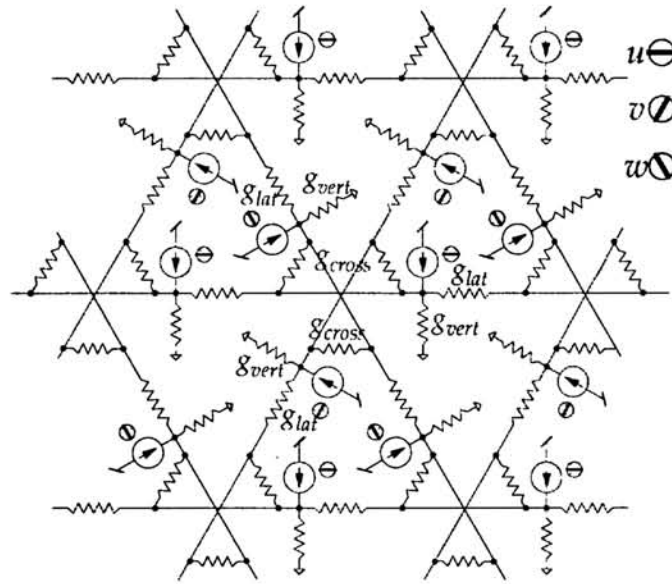

Figure 3: *Network of bipole cells, implemented on a hexagonal resistive grid using orientationally tuned diffusors extending in three directions. $g_{lat}/g_{vert}$ determines the spatial extent of the dipole, whereas $g_{lat}/g_{cross}$ sets the directional selectivity.*

The bipole cell resistive grid (Figure 3) implements a three-fold cross-coupled, directionally polarized, long-range diffusive kernel, formulated as follows:

$$B_0^u = K_u^u C_0^u + K_v^u C_0^v + K_w^u C_0^w \tag{2}$$

where $K_u^u$, $K_v^u$, and $K_w^u$ represent spatial convolutional kernels implementing bipole fields symmetrically polarized in the $u$, $v$ and $w$ directions. Diffusive kernels can be efficiently implemented with a distributed representation using resistive diffusive elements [7, 10]. Three linear networks of diffusor elements are used, complemented with cross-links of adjustable strength, to control the degree of direction selectivity and the spatial spread of the kernel. Finally, the result (2) is locally normalized, before it is fed back onto the complex cells.

## 3   Analog VLSI Implementation

The simplified circuit diagram of the BCS cell, including simple, complex and bipole cell functions on a hexagonal grid, is shown in Figure 4.

The image is acquired either optically from phototransistors on the focal-plane, or in direct electronic format through random-access pixel addressing, Figure 4 (a). The simple cell portion in Figure 4 (b) combines the local intensity $I_0$ with intensities $I_v$ and $I_w$ received from neighboring cells to compute the rectified gradient in (1), using distributed current mirrors and an absolute value circuit. A pMOS load converts the complex cell output into a voltage representation $C_0^u$ for distribution to neighboring nodes and complementary orientations: local inhibition for spatial and directional competition in Figure 4 (c), and long-range cooperation through the bipole layer in Figure 4 (d). The linear diffusive kernel is implemented in current-mode using ladder structures of subthreshold MOS transistors [7], three families extending in each direction with cross-links for directional dispersion as indicated in Figure 3.

Voltage biases control the spatial extent and directional selectivity of the interactions, as

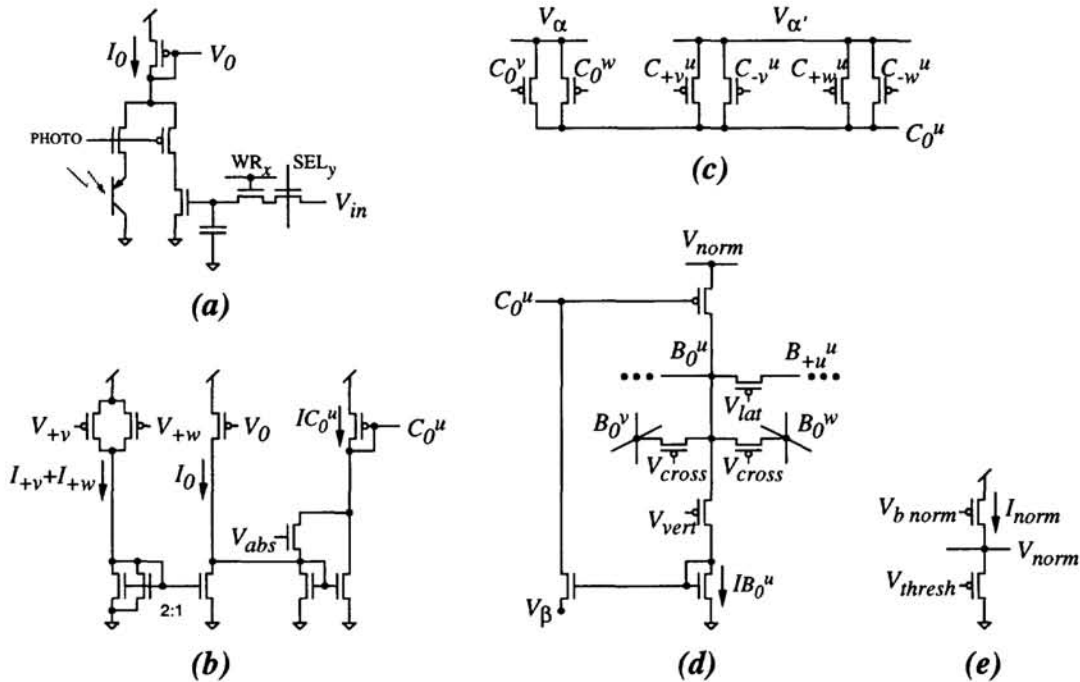

Figure 4: *Simplified circuit schematic of one BCS cell in the hexagonal array, showing only one of three directions, the other directions being symmetrical in implementation. (a) Photosensor and random-access input selection circuit. (b) Simple cell rectified gradient calculation. (c) Complex cell spatial and orientational inhibition. (d) Bipole cell directional long range cooperation. (e) Bipole global gain and threshold control.*

well as the relative strength of inhibition and excitation, and the level of renormalization, for the complex and bipole cells. The values for $g_{vert}$, $g_{lat}$ and $g_{cross}$ controlling the bipole kernel are set externally by applying gate bias voltages $V_{vert}$, $V_{lat}$ and $V_{cross}$, respectively. Likewise, the constants $\alpha$, $\alpha'$ and $\beta$ in (1) are set independently by the applied source voltages $V_\alpha$, $V_{\alpha'}$ and $V_\beta$. Global normalization and thresholding of the bipole response for improved stability of edge formation is achieved through an additional diffusive network that acts as a localized Gilbert-type current normalizer (only partially shown in Figure 4 (e)).

## 4 Experimental Results

A prototype $12 \times 10$ pixel array has been fabricated and tested. The pixel unit, illustrated in Figure 5 (a), has been designed for testability, and has not been optimized for density. The pixel contains 88 transistors including a phototransistor, a large sample-and-hold capacitor, and three networks of interconnections in each of the three directions, requiring a fan-in/fan-out of 18 node voltages across the interface of each pixel unit. A micrograph of the Tiny $2.2 \times 2.2$ sq. mm chip, fabricated through MOSIS in 1.2 $\mu$m CMOS technology, is shown in Figure 5 (b).

We have tested the BCS chip both under focal-plane optical inputs, and random-access direct electronic inputs. Input currents from optical input under ambient room lighting conditions are around 30 nA. The experimental results reported here are obtained by feeding test inputs electronically. The response of the BCS chip to two test images of interest are shown in Figures 6 and 7.

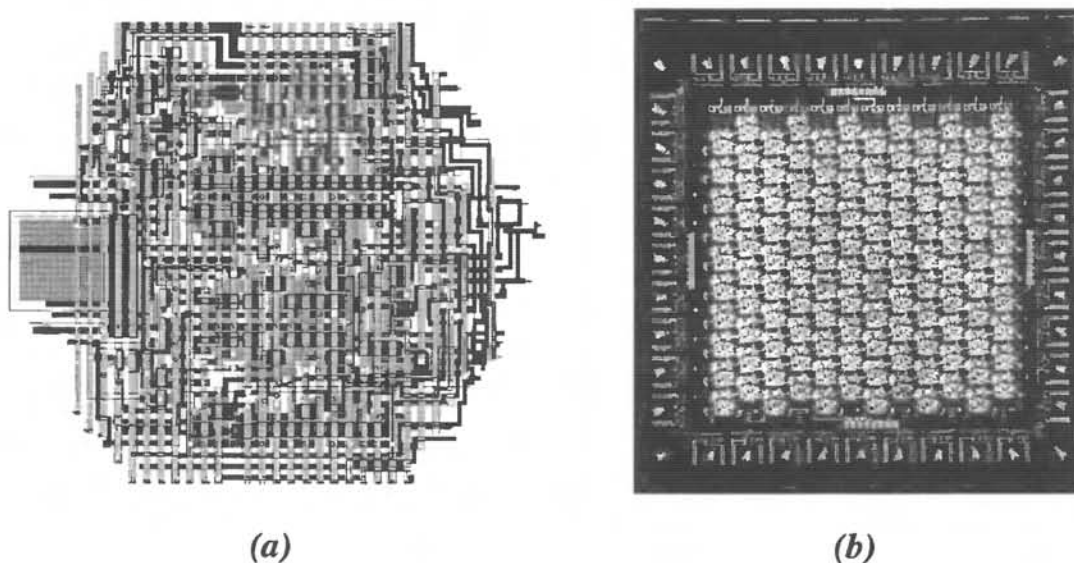

*(a)*                                                                           *(b)*

Figure 5: *BCS processor.   (a) Pixel layout.   (b) Chip micrograph.*

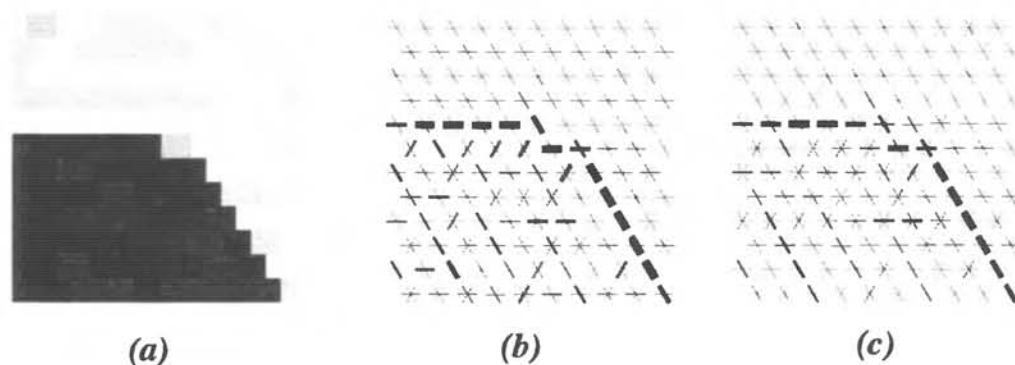

*(a)*                                  *(b)*                                  *(c)*

Figure 6:  *Experimental response of the BCS chip to a curved edge.    (a) Reconstructed
input image.   (b) Complex field.   (c) Bipole field.    The thickness of the bars on the grid
represent the measured components in the three directions.*

Figure 6 illustrates the interpolating directional response to a curved edge in the input, vary-
ing in direction between two of the principal axes ($u$ and $w$ in the example). Interpolation
between quantized directions is important since implementing more axes on the grid incurs
a quadratic cost in complexity. The second example image contains a bar with two gaps of
different diameter, for the purpose of testing BCS's capacity to extend contour boundaries
across clutter. The response in Figure 7 illustrates a characteristic of bipole operation, in
which short-range discontinuities are bridged but large ones are preserved.

## 5   Conclusions

An analog VLSI cellular architecture implementing the Boundary Contour System (BCS)
on the focal plane has been presented. A diffusive kernel with distributed resistive networks
has been used to implement long-range interactions of bipole cells without the need of
excessive global interconnects across the array of pixels. The cellular model is fairly easy to
implement, and succeeds in selecting boundary contours in images with significant clutter.

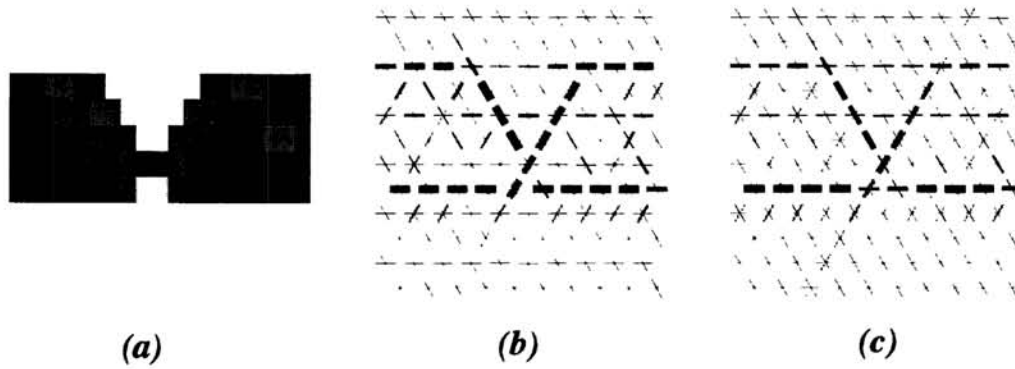

*(a)*                          *(b)*                          *(c)*

Figure 7:  *Experimental response of the BCS chip to a bar with two gaps of different size.*
*(a) Reconstructed input image.   (b) Complex field.   (c) Bipole field.*

Experimental results from a 12 × 10 pixel prototype demonstrate expected BCS operation
on simple examples. While this size is small for practical applications, the analog cellular
architecture is fully scalable towards higher resolutions. Based on the current design, a
10, 000-pixel array in 0.5 $\mu$m CMOS technology would fit a 1 cm$^2$ die.

## Acknowledgments

This research was supported by DARPA and ONR under MURI grant N00014-95-1-0409.
Chip fabrication was provided through the MOSIS service.

## References

[1]  S. Grossberg, "Neural Networks for Visual Perception in Variable Illumination," *Optics News*,
pp. 5–10, August 1988.

[2]  S. Grossberg, "A Solution of the Figure-Ground Problem for Biological Vision," *Neural Net-
works*, vol. **6**, pp. 463–482, 1993.

[3]  S. Grossberg, E. Mingolla, and J. Williamson, "Synthetic Aperture Radar Processing by a
Multiple Scale Neural System for Boundary and Surface Representation," *Neural Networks*,
vol. **9** (1), January 1996.

[4]  Z.P. Li, "A Neural Model of Contour Integration in the Primary Visual Cortex," *Neural Com-
putation*, vol. **10** (4), pp. 903-940, 1998.

[5]  K.A. Boahen, "A Retinomorphic Vision System," *IEEE Micro*, vol. **16** (5), pp. 30-39, Oct.
1996.

[6]  J.G. Harris, C. Koch, and J. Luo, "A Two-Dimensional Analog VLSI Circuit for Detecting
Discontinuities in Early Vision," *Science*, vol. **248**, pp. 1209-1211, June 1990.

[7]  A.G. Andreou, K.A. Boahen, P.O. Pouliquen, A. Pavasovic, R.E. Jenkins, and K. Strohbehn,
"Current-Mode Subthreshold MOS Circuits for Analog VLSI Neural Systems," *IEEE Transac-
tions on Neural Networks*, vol. **2** (2), pp 205-213, 1991.

[8]  L. Dron McIlrath, "A CCD/CMOS Focal-Plane Array Edge Detection Processor Implementing
the Multiscale Veto Algorithm," *IEEE J. Solid State Circuits*, vol. **31** (9), pp 1239-1248, 1996.

[9]  P. Venier, A. Mortara, X. Arreguit and E.A. Vittoz, "An Integrated Cortical Layer for Orienta-
tion Enhancement," *IEEE J. Solid State Circuits*, vol. **32** (2), pp 177-186, Febr. 1997.

[10]  E. Fragniere, A. van Schaik and E. Vittoz, "Reactive Components for Pseudo-Resistive Net-
works," Electronic Letters, vol. **33** (23), pp 1913-1914, Nov. 1997.